# Assessing the Quality of Learned Local Models

**Stefan Schaal**          **Christopher G. Atkeson**
Department of Brain and Cognitive Sciences & The Artifical Intelligence Laboratory
Massachusetts Institute of Technology
545 Technology Square, Cambridge, MA 02139
email: sschaal@ai.mit.edu, cga@ai.mit.edu

## Abstract

An approach is presented to learning high dimensional functions in the case where the learning algorithm can affect the generation of new data. A local modeling algorithm, locally weighted regression, is used to represent the learned function. Architectural parameters of the approach, such as distance metrics, are also localized and become a function of the query point instead of being global. Statistical tests are given for when a local model is good enough and sampling should be moved to a new area. Our methods explicitly deal with the case where prediction accuracy requirements exist during exploration: By gradually shifting a "center of exploration" and controlling the speed of the shift with local prediction accuracy, a goal-directed exploration of state space takes place along the fringes of the current data support until the task goal is achieved. We illustrate this approach with simulation results and results from a real robot learning a complex juggling task.

## 1    INTRODUCTION

Every learning algorithm faces the problem of sparse data if the task to be learned is sufficiently nonlinear and high dimensional. Generalization from a limited number of data points in such spaces will usually be strongly biased. If, however, the learning algorithm has the ability to affect the creation of new experiences, the need for such bias can be reduced. This raises the questions of (1) how to sample data the most efficient, and (2) how to assess the quality of the sampled data with respect to the task to be learned. To address these questions, we represent the task to be learned with local linear models. Instead of constraining the number of linear models as in other approaches, infinitely many local models are permitted. This corresponds to modeling the task with the help of (hyper-) tangent planes at every query point instead of representing it in a piecewise linear fashion. The algorithm applied for this purpose, locally weighted regression (LWR), stems from nonparametric regression analysis (Cleveland, 1979, Müller, 1988, Härdle 1990, Hastie&Tibshirani, 1991). In Section 2, we will briefly outline LWR. Section 3 discusses

several statistical tools for assessing the quality of a learned linear LWR model, how to optimize the architectural parameters of LWR, and also how to detect outliers in the data. In contrast to previous work, all of these statistical methods are local, i.e., they depend on the data in the proximity of the current query point and not on all the sampled data. A simple exploration algorithm, the shifting setpoint algorithm (SSA), is used in Section 4 to demonstrate how the properties of LWR can be exploited for learning control. The SSA explicitly controls prediction accuracy during learning and samples data with the help of optimal control techniques. Simulation results illustrate that this method work well in high dimensional spaces. As a final example, the methods are applied to a real robot learning a complex juggling task in Section 5.

## 2    LOCALLY WEIGHTED REGRESSION

Locally linear models constitute a good compromise between locally constant models such as nearest neighbors or moving average and locally higher order models; the former tend to introduce too much bias while the latter require fitting many parameters which is computationally expensive and needs a lot of data. The algorithm which we explore here, locally weighted regression (LWR) (Atkeson, 1992, Moore, 1991, Schaal&Atkeson, 1994), is closely related to versions suggested by Cleveland et al. (1979, 1988) and Farmer&Siderowich (1987). A LWR model is trained by simply storing every experience as an input/output pair in memory. If an output $y_q$ is to be generated from a given input $\mathbf{x}_q$, the it is computed by fitting a (hyper-) tangent plane at $\mathbf{x}_q$ by means of weighted regression:

$$\beta(\mathbf{x}_q) = \left[(\mathbf{WX})^T \mathbf{WX}\right]^{-1} (\mathbf{WX})^T \mathbf{Wy} \tag{1}$$

where $\mathbf{X}$ is an $m \times (n+1)$ matrix of inputs to the regression, $\mathbf{y}$ the vector of corresponding outputs, $\beta(\mathbf{x}_q)$ the vector of regression parameters, and $\mathbf{W}$ the diagonal $m \times m$ matrix of weights. The requested $y_q$ results from evaluating the tangent plane at $\mathbf{x}_q$, i.e., $y_q = \mathbf{x}_q^T \beta$. The elements of $\mathbf{W}$ give points which are close to the current query point $\mathbf{x}_q$ a larger influence than those which are far away. They are determined by a Gaussian kernel:

$$w_i(\mathbf{x}_q) = \exp\left((\mathbf{x}_i - \mathbf{x}_q)^T \mathbf{D}(\mathbf{x}_q)(\mathbf{x}_i - \mathbf{x}_q) / 2k(\mathbf{x}_q)^2\right) \tag{2}$$

$w_i$ is the weight for the $i^{th}$ data point $(\mathbf{x}_i, \mathbf{y}_i)$ in memory given query point $\mathbf{x}_q$. The matrix $\mathbf{D}(\mathbf{x}_q)$ weights the contribution of the individual input dimensions, and the factor $k(\mathbf{x}_q)$ determines how local the regression will be. $\mathbf{D}$ and $k$ are architectural parameters of LWR and can be adjusted to optimize the fit of the local model. In the following we will just focus on optimizing $k$, assuming that $\mathbf{D}$ normalizes the inputs and needs no further adjustment; note that, with some additional complexity, our methods would also hold for locally tuning $\mathbf{D}$.

## 3    ASSESSING THE LOCAL FIT

In order to measure the goodness of the local model, several tests have been suggested. The most widely accepted one is leave-one-out cross validation (CV) which calculates the prediction error of every point in memory after recalculating (1) without this point (Wahba&Wold 1975, Maron&Moore 1994). As an alternative measure, Cleveland et al. (1988) suggested Mallow's $C_p$-test, originally developed as a way to select covariates in linear regression analysis (Mallow, 1966). Hastie&Tibshirani (1991) showed that CV and the $C_p$-test are closely related for certain classes of analyses. Hastie&Tibshirani (1991)

also presented pointwise standard-error bands to assess the confidence in a fitted value which correspond to confidence bands in the case of an unbiased fit. All these tests are essentially global by requiring statistical analysis over the entire range of data in memory. Such a global analysis is computationally costly, and it may also not give an adequate measure at the current query site $\mathbf{x}_q$: the behavior of the function to be approximated may differ significantly in different places, and an averaging over all these behaviors is unlikely to be representative for all query sites (Fan&Gijbels, 1992).

It is possible to convert some of the above measures to be local. Global cross validation has a relative in linear regression analysis, the PRESS residual error (e.g., Myers, 1990), here formulated as a mean squared local cross validation error:

$$MSE_{cross}(\mathbf{x}_q) = \frac{1}{n'-p'} \sum_{i=1}^{n} \left( \frac{w_i(y_i - \mathbf{x}_i^T \beta)}{1 - w_i \mathbf{x}_i^T (\mathbf{X}^T \mathbf{W}^T \mathbf{W} \mathbf{X})^{-1} \mathbf{x}_i w_i} \right)^2, \quad n' = \sum_{i=1}^{n} w_i^2, \quad p' = \frac{n' p}{n} \quad (3)$$

$n$ is the number of data points in memory contributing with a weight $w_i$ greater than some small constant (e.g., $w_i > 0.01$) to the regression, and $p$ is the dimensionality of $\beta$. The PRESS statistic performs leave-one-out cross validation computationally very efficient by not requiring the recalculation of $\beta$ (Eq.(1)) for every excluded point.

Analogously, prediction intervals from linear regression analysis (e.g., Myers, 1990) can be transformed to be a local measure too:

$$I_q = \mathbf{x}_q^T \beta \pm t_{\alpha/2,n'-p'} \, s\sqrt{1 + \mathbf{x}_q^T (\mathbf{X}^T \mathbf{W}^T \mathbf{W} \mathbf{X})^{-1} \mathbf{x}_q} \quad (4)$$

where $s^2$ is an estimate of the variance at $\mathbf{x}_q$:

$$s^2(\mathbf{x}_q) = \frac{(\mathbf{X}\beta - \mathbf{y})^T \mathbf{W}^T \mathbf{W}(\mathbf{X}\beta - \mathbf{y})}{n' - p'} \quad (5)$$

and $t_{\alpha/2,n'-p'}$ is Student's t-value of $n'-p'$ degrees of freedom for a $100(1-\alpha)\%$ prediction bound. The direct interpretation of (4) as prediction bounds is only possible if $y_q$ is an unbiased estimate, which is usually hard to determine.

Finally, the PRESS statistic can also be used for local outlier detection. For this purpose it is reformulated as a standardized individual PRESS residual:

$$e_{i,cross}(\mathbf{x}_q) = \frac{w_i(y_i - \mathbf{x}_i^T \beta)}{s\sqrt{1 - w_i \mathbf{x}_i^T (\mathbf{X}^T \mathbf{W}^T \mathbf{W} \mathbf{X})^{-1} \mathbf{x}_i w_i}} \quad (6)$$

This measure has zero mean and unit variance. If it exceeds a certain threshold for a point $\mathbf{x}_i$, the point can be called an outlier.

An important ingredient to forming the measures (3)-(6) lies in the definition of $n'$ and $p'$ as given in (3). Imagine that the weighting function (2) is not Gaussian but rather a function that clips data points whose distance from the current query point exceeds a certain threshold and that the remaining $r$ data points all contribute with unit weight. This reduced data regression coincides correctly with a $r$-data regression since $n' = r$. In the case of the soft-weighting (2), the definition of $n'$ ensures the proper definition of the moments of the data. However, the definition of $p'$, i.e., the degrees of freedom of the regression, is somewhat arbitrary since it is unclear how many degrees of freedom have ac-

tually been used. Defining $p'$ as in (3) guarantees that $p' < n'$ and renders all results more pessimistic when only a small number of data points contribute to the regression.

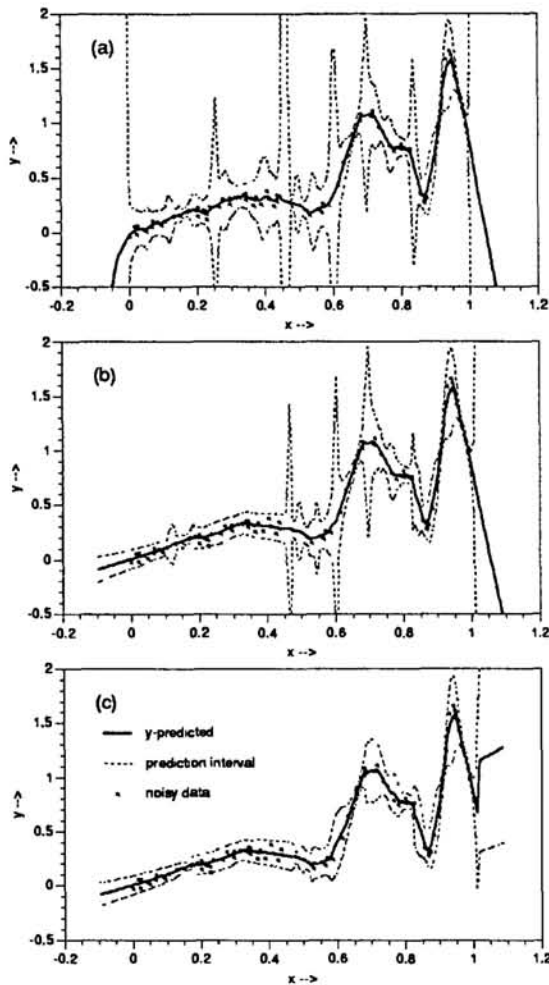

Figure 1: Optimizing the LWR fit using: (a) global cross validation; (b) local cross validation; (c) local prediction intervals.

The statistical tests (3) and (4) can not only be used as a diagnostic tool, but they can also serve to optimize the architectural parameters of LWR. This results in a function fitting technique which is called supersmoothing in statistics (Hastie&Tibshirani, 1991). Fan&Gijbels (1992) investigated a method for this purpose that required estimation of the second derivative of the function to be approximated and the data density distribution. These two measures are not trivially obtained in high dimensions and we would like to avoid using them. Figure 1 shows fits of noisy data from the function $y = x - \sin^3(2\pi x^3) \cos(2\pi x^3) \exp(x^4)$ with 95% prediction intervals around the fitted values. In Figure 1a, global one-leave-out cross validation was applied to optimize $k$ (cf. Eq.(2)). In the left part of the graph the fit starts to follow noise. Such behavior is to be expected since the global optimization of $k$ also took into account the quickly changing regions on the right side of the graph and thus chose a rather small $k$. In Figure 1b minimization of the local one-leave-out cross validation error was applied to fit the data, and in Figure 1c prediction intervals were minimized. These two fits cope nicely with both the high frequency and the low frequency regions of the data and recover the true function rather well. The extrapolation properties of local cross validation are the most appropriate given that the we know the true function.

Interestingly, at the right end of Figure 1c, the minimization of the prediction intervals suddenly detects that global regression has a lower prediction interval than local regression and jumps into the global mode by making $k$ rather large. In both local methods there is always a competition between local and global regression. But sudden jumps take place only when the prediction interval is so large that the data is not trustworthy anyway.

To some extend, the statistical tests (3)-(6) implicitly measure the data density at the current query point and are thus sensitive towards little data support, characterized by a small $n'$. This property is desirable as a diagnostic tool, particularly if the data sampling process can be directed towards such regions. However, if a fixed data set is to be analyzed which has rather sparse and noisy data in several regions, a fit of the data with local optimization methods may result in too jagged an approximation since the local fitting mistakes the noise in such regions as high frequency portion of the data. Global methods avoid this effect by biasing the function fitting in such unfavorable areas with knowledge from other data regions and will produce better results if this bias is appropriate.

## 4    THE SHIFTING SETPOINT EXPLORATION ALGORITHM

In this section we want to give an example of how LWR and its statistical tools can be used for goal directed data sampling in learning control. If the task to be learned is high dimensional it is not possible to leave data collection to random exploration; on the one hand this would take too much time, and on the other hand it may cause the system to enter unsafe or costly regions of operation. We want to develop an exploration algorithm which explicitly avoids with such problems. The shifting setpoint algorithm (SSA) attempts to decompose the control problem into two separate control tasks on different time scales. At the fast time scale, it acts as a nonlinear regulator by trying to keep the controlled system at some chosen setpoints in order to increase the data density at these setpoints. On a slower time scale, the setpoints are shifted by controlling local prediction accuracy to accomplish a desired goal. In this way the SSA builds a narrow tube of data support in which it knows the world. This data can be used by more sophisticated control algorithms for planning or further exploration.

The algorithm is graphically illustrated in the example of a mountain car in Figure 2. The task of the car is to drive at a given constant *horizontal* speed $\dot{x}_{desired}$ from the left to the right of Figure 2a. $\dot{x}_{desired}$ need not be met precisely; the car should also minimize its fuel consumption. Initially, the car knows nothing about the world and cannot look ahead, but it has noisy feedback of its position and velocity. Commands, which correspond to the thrust $F$ of the motor, can be generated at 5Hz. The mountain car starts at its start point with one arbitrary initial action for the first time step; then it brakes and starts all over again, assuming the system can be reset somehow. The discrete one step dynamics of the car are modeled by an LWR forward model:

$$\hat{x}_{next} = \hat{f}(x_{current}, F), \quad where \quad x = (\dot{x}, x)^T \tag{7}$$

After a few trials, the SSA searches the data in memory for the point $(x_{current}^T, F, x_{next}^T)_{best}^T$ whose outcome $\hat{x}_{next}$ can be predicted with the smallest local prediction interval. This best point is declared the setpoint of this stage:

$$(x_{S,in}^T, F_S, x_{S,out}^T)^T = (x_{current}^T, F, \hat{x}_{next}^T)_{best}^T \tag{8}$$

and its local linear model results from a corresponding LWR lookup:

$$x_{S,out} = \hat{f}(x_{S,in}, F_S) \approx Ax_{S,in} + BF_S + c \tag{9}$$

Based on this linear model, an optimal LQ controller (e.g., Dyer&McReynolds, 1970) can be constructed. This results in a control law of the form:

$$F^* = -K(x_{current} - x_{S,in}) + F_S. \tag{10}$$

After these calculations, the mountain car learned one controlled action for the first time step. However, since the initial action was chosen arbitrarily, $x_{S,out}$ will be significantly away from the desired speed $\dot{x}_{desired}$. A reduction of this error is achieved as follows. First, the SSA repeats one step actions with the LQ controller until sufficient data is collected to reduce the prediction intervals of LWR lookups for $(x_{S,in}^T, F_S)^T$ (Eq.(9)) below a certain threshold. Then it shifts the setpoint towards the goal according to the procedure:

1) calculate the error of the predicted output state:     $err_{S,out} = x_{desired} - x_{S,out}$
2) take the derivative of the error with respect to the command $F_S$ from a LWR lookup for $(x_{S,in}^T, F_S)^T$ (cf. (9)):

$$\frac{\partial \mathbf{err}_{S,out}}{\partial F_S} = \frac{\partial \mathbf{err}_{S,out}}{\partial x_{S,out}} \frac{\partial x_{S,out}}{\partial F_S} = -\frac{\partial x_{S,out}}{\partial F_S} = -\mathbf{B}$$

and calculate a correction $\Delta F_S$ from solving: $-\mathbf{B}\Delta F_S = \alpha \, \mathbf{err}_{S,out}$; $\alpha \in [0,1]$ determines how much of the error should be compensated for in one step.

3) update $F_S$: $F_S = F_S - \Delta F_S$ and calculate the new $\mathbf{x}_{S,out}$ with LWR (Eq.(9)).
4) assess the fit for the updated setpoint with prediction intervals. If the quality is above a certain threshold, continue with 1), otherwise terminate shifting.

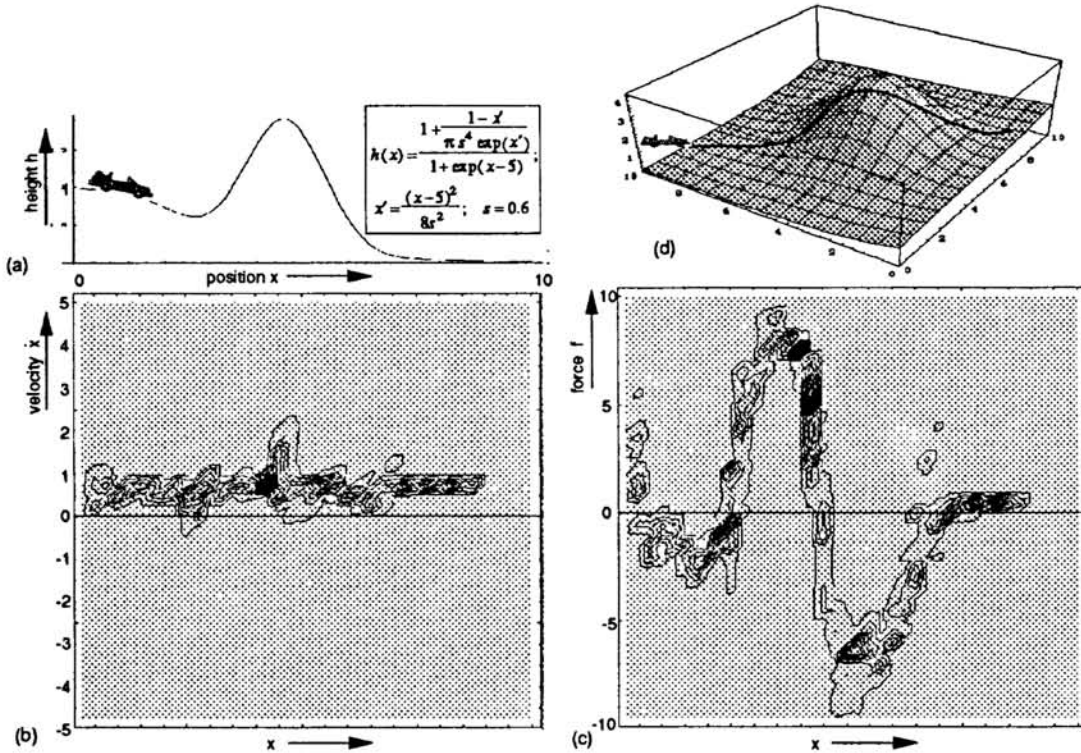

Figure 2: The mountain car: (a) landscape across which the car has to drive at constant velocity of 0.8 m/s, (b) contour plot of data density in phase space as generated by using multistage SSA, (c) contour plot of data density in position-action space, (d) 2-dimensional mountain car

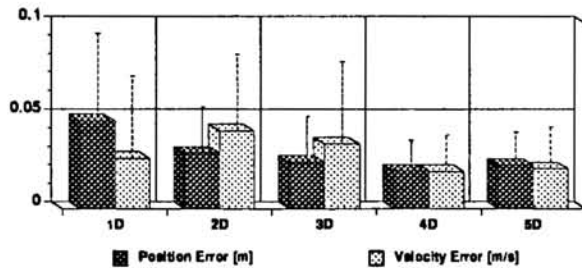

Figure 3: Mean prediction error of local models

In this way, the output state of the setpoint shifts towards the goal until the data support falls below a threshold. Now the mountain car performs several new trials with the new setpoint and the correspondingly updated LQ controller. After the quality of fit statistics rise above a threshold, the setpoint can be shifted again. As soon as the first stage's setpoint reduces the error $\mathbf{x}_{desired} - \mathbf{x}_{S,out}$ sufficiently, a new stage is created and the mountain car tries to move one step further in its world. The entire procedure is repeated for each new stage until the car knows how to move across the landscape. Figure 2b and Figure 2c show the thin band of data which the algorithm collected in state space and position-action space. These two pictures together form a narrow tube of knowledge in the input space of the forward model.

The example of the mountain car can easily be scaled up to arbitrarily high dimensions by making the mountain a multivariate function. We tried versions up to a 5-dimensional mountain corresponding to a $\Re^{15} \rightarrow \Re^{10}$ forward model; Figure 2d shows the 2-dimensional version. The results of learning had the same quality as in the 1D example. Figure 3 shows the prediction errors of the local models after learning for the 1D, 2D,..., and 5D mountain car. To obtain these errors, the car was started at random positions within its data support from where it drove along the desired trajectory. The difference between the predicted next state and the actual outcome at each time step was averaged. Position errors stayed within 2-4 cm on the 10m long landscape, and velocity errors within 0.02-0.05 m/s. The dimensionality of the problem did not affect the outcome significantly.

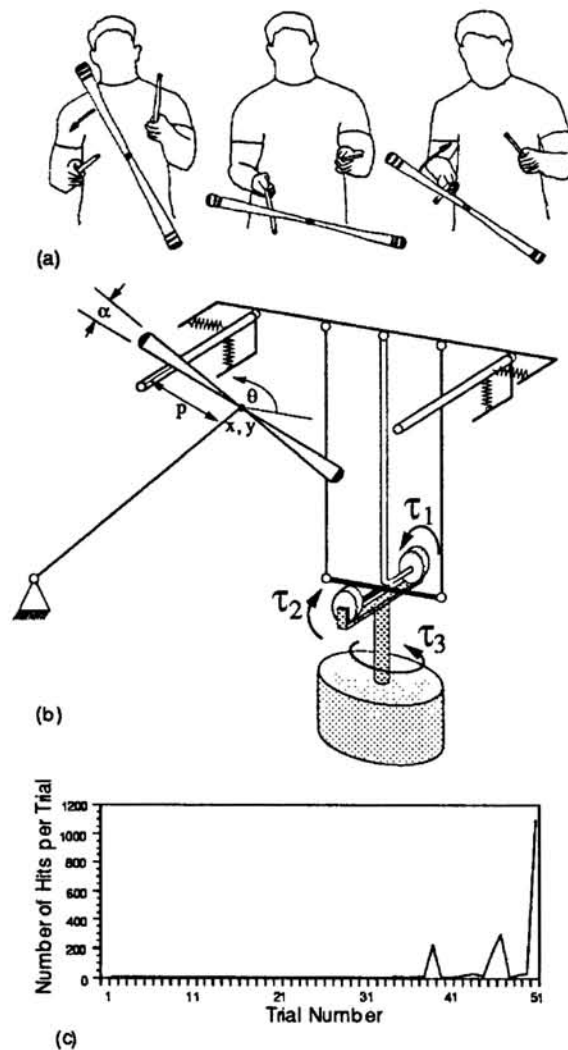

(a)

(b)

(c)

Figure 4: (a) illustration of devilsticking, (b) a devilsticking robot, (c) learning curve of robot

## 5     ROBOT JUGGLING

To test our algorithms in a real world experiment, we implemented them on a juggling robot. The juggling task to be performed, devil sticking, is illustrated in Figure 4a. For the robot, devil sticking was slightly simplified by attaching the devil stick to a boom, as illustrated in Figure 4b. The task state was encoded as a 5-dimensional state vector, taken at the moment when the devilstick hit one of the hand sticks; the throw action was parameterized as 5-dimensional action vector. This resulted in a $\Re^{10} \rightarrow \Re^5$ discrete forward model of the task. Initially the robot was given default actions for the left-hand and right-hand throws; the quality of these throws, however, was far away from achieving steady juggling. The robot started with no initial experiences and tried to build controllers to perform continuous juggling. The goal states for the SSA developed automatically from the requirement that the left hand had to learn to throw the devilstick to a place where the right hand had sufficient data support to control the devilstick, and vice versa. Figure 4c shows a typical learning curve for this task. It took about 40 trials before the left and the right hand learned to throw the devilstick such that both hands were able to cooperate. Then, performance quickly went up to long runs up to 1200 consecutive hits. Humans usually need about one week of one hour practicing per day before they achieve decent juggling performance. In comparison to this, the learning algorithm performed very well. However, it has to be pointed out that the learned controllers were only local and could not cope with larger perturbations. A detailed description of this experiment can be found in Schaal&Atkeson (1994).

# CONCLUSIONS

One of the advantages of memory-based nonparametric learning methods lies in the least commitment strategy which is associated with them. Since all data is kept in memory, a lookup can be optimized with respect to the architectural parameters. Parametric approaches do not have this ability if they discard their training data; if they retain it, they essentially become memory-based. The origin of nonparametric modeling in traditional statistics provides many established statistical methods to inspect the quality of what has been learned by the system. Such statistics formed the backbone of the SSA exploration algorithm. So far we have only examined some of the most obvious statistical tools which directly relate to regression analysis. Many other methods from other statistical frameworks may be suitable as well and will be explored by our future work.

## Acknowledgements

Support was provided by the Air Force Office of Scientific Research, by the Siemens Corporation, the German Scholarship Foundation and the Alexander von Humboldt Foundation to Stefan Schaal, and a National Science Foundation Presidential Young Investigator Award to Christopher G. Atkeson. We thank Gideon Stein for implementing the first version of LWR on a DSP board, and Gerrie van Zyl for building the devil sticking robot and implementing the first version of learning of devil sticking.

## References

Atkeson, C.G. (1992), "Memory-Based Approaches to Approximating Continuous Functions", in: Casdagli, M.; Eubank, S. (eds.): *Nonlinear Modeling and Forecasting*. Redwood City, CA: Addison Wesley (1992).

Cleveland, W.S., Devlin, S.J., Grosse, E. (1988), "Regression by Local Fitting: Methods, Properties, and Computational Algorithms". *Journal of Econometrics* 37, 87-114, North-Holland (1988).

Cleveland, W.S. (1979), "Robust Locally-Weighted Regression and Smoothing Scatterplots". *Journal of the American Statistical Association* , no.74, pp.829-836 (1979).

Dyer, P., McReynolds, S.R. (1970), *The Computation and Theory of Optimal Control*, New York: Academic Press (1970).

Fan, J., Gijbels, I. (1992), "Variable Bandwidth And Local Linear Regression Smoothers", *The Annals of Statistics*, vol.20, no.4, pp.2008-2036 (1992).

Farmer, J.D., Sidorowich, J.J. (1987), "Predicting Chaotic Dynamics", Kelso, J.A.S., Mandell, A.J., Shlesinger, M.F., (eds.):*Dynamic Patterns in Complex Systems*, World Scientific Press (1987).

Härdle, W. (1991), *Smoothing Techniques with Implementation in S*, New York, NY: Springer.

Hastie, T.J.; Tibshirani, R.J. (1991), *Generalized Additive Models*, Chapman and Hall.

Mallows, C.L. (1966), "Choosing a Subset Regression", unpublished paper presented at the annual meeting of the American Statistical Association, Los Angles (1966).

Maron, O., Moore, A.W. (1994), "Hoeffding Races: Accelerating Model Selection Search for Classification and Function Approximation", in: Cowan, J. , Tesauro, G., and Alspector, J. (eds.) *Advances in Neural Information Processing Systems 6*, Morgan Kaufmann (1994).

Müller, H.-G. (1988), *Nonparametric Regression Analysis of Longitudinal Data*, Lecture Notes in Statistics Series, vol.46, Berlin: Springer (1988).

Myers, R.H. (1990), *Classical And Modern Regression With Applications*, PWS-KENT (1990).

Schaal, S., Atkeson, C.G. (1994), "Robot Juggling: An Implementation of Memory-based Learning", to appear in:*Control Systems Magazine*, Feb. (1994).

Wahba, G., Wold, S. (1975), "A Completely Automatic French Curve: Fitting Spline Functions By Cross-Validation", *Communications in Statistics*, 4(1) (1975).